# Empirical models of spiking in neural populations

**Jakob H. Macke**
Gatsby Computational Neuroscience Unit
University College London, UK
jakob@gatsby.ucl.ac.uk

**Lars Büsing**
Gatsby Computational Neuroscience Unit
University College London, UK
lars@gatsby.ucl.ac.uk

**John P. Cunningham**
Department of Engineering
University of Cambridge, UK
jpc74@cam.ac.uk

**Byron M. Yu**
ECE and BME
Carnegie Mellon University
byronyu@cmu.edu

**Krishna V. Shenoy**
Department of Electrical Engineering
Stanford University
shenoy@stanford.edu

**Maneesh Sahani**
Gatsby Computational Neuroscience Unit
University College London, UK
maneesh@gatsby.ucl.ac.uk

## Abstract

Neurons in the neocortex code and compute as part of a locally interconnected population. Large-scale multi-electrode recording makes it possible to access these population processes empirically by fitting statistical models to unaveraged data. What statistical structure best describes the concurrent spiking of cells within a local network? We argue that in the cortex, where firing exhibits extensive correlations in both time and space and where a typical sample of neurons still reflects only a very small fraction of the local population, the most appropriate model captures shared variability by a low-dimensional latent process evolving with smooth dynamics, rather than by putative direct coupling. We test this claim by comparing a latent dynamical model with realistic spiking observations to coupled generalised linear spike-response models (GLMs) using cortical recordings. We find that the latent dynamical approach outperforms the GLM in terms of goodness-of-fit, and reproduces the temporal correlations in the data more accurately. We also compare models whose observations models are either derived from a Gaussian or point-process models, finding that the non-Gaussian model provides slightly better goodness-of-fit and more realistic population spike counts.

## 1 Introduction

Multi-electrode array recording and similar methods provide measurements of activity from dozens of neurons simultaneously, and thus allow unprecedented insights into the statistical structure of neural population activity. To exploit this potential we need methods that identify the temporal dynamics of population activity and link it to external stimuli and observed behaviour. These statistical models of population activity are essential for understanding neural coding at a population level [1] and can have practical applications for Brain Machine Interfaces [2].

Two frameworks for modelling the temporal dynamics of cortical population recordings have recently become popular. Generalised Linear spike-response Models (GLMs) [1, 3, 4, 5] model the influence of spiking history, external stimuli or other neural signals on the firing of a neuron. Here, the interdependence of different neurons is modelled by terms that link the instantaneous firing rate of each neuron to the recent spiking history of the population. The parameters of the GLM can be

learned efficiently by convex optimisation [3, 4, 5, 6]. Such models have been successful in a range of studies and systems, including retinal [1] and cortical [7] population recordings.

An alternative is provided by latent variable models such as Gaussian Process Factor Analysis [8] or other state-space models [9, 10, 11]. In this approach, shared variability (or 'noise correlation') is modelled by an unobserved process driving the population, which is sometimes characterised as 'common input' [12, 13]. One advantage of this approach is that the trajectories of the latent state provide a compact, low-dimensional representation of the population which can be used to visualise population activity, and link it to observed behaviour [14].

## 1.1 Comparing coupled generalised linear models and latent variable models

Three lines of argument suggest that latent dynamical models may provide a better fit to cortical population data than the spike-response GLM. First, prevalent recording apparatus, such as extracellular grid electrodes, sample neural populations very sparsely making it unlikely that much of the observed shared variability is a consequence of direct physical interaction. Hence, the coupling filters of a GLM rather reflect statistical interactions (sometimes called *functional connectivity*). Without direct synaptic coupling, it is unlikely that variability is shared exclusively by particular pairs of units; instead, it will generally be common to many cells—an assumption explicit in the latent variable approach, where shared variability results from the model of cortical dynamics.

Second, most cortical population recordings find that shared variability across neurons is dominated by a central peak at zero time lag (i.e. the strongest correlation is instantaneous) [15, 16], and has broad, positive, sometimes asymmetric flanks, decaying slowly with lag time. Correlations with these properties arise naturally in dynamical system models. The common input from the latent state induces instantaneous correlations, and the evolution of the latent system typically yields positive temporal correlations over moderate timescales. By contrast, GLMs couple instantaneous rate to the *recent* spiking of other neurons, but not to their *simultaneous* activity, making zero-lag correlation hard to model. (As we show below in "Methods", the inclusion of simultaneous terms would lead to invalid models.) Instead, the common approach is to discretise time very finely so that an off-zero peak can be brought close to simultaneity. This increases computational load, and often requires discretisation finer than the time-scale of interest, perhaps even finer than the recording resolution (e.g. for 2-photon calcium imaging). In addition, positive history coupling in a GLM may lead to loops of self-excitation, predicting unrealistically high firing rates—a trend that must be countered by long-term negative self-coupling. Thus, while it is certainly not impossible to reproduce neural correlation structure with GLMs [1], they do not seem to be the natural choice for modelling time-series of spike-counts with instantaneous correlations.

Third, recording time, and therefore the data available to fit a model, is usually limited *in vivo*, especially in behaving animals. This paucity of data places strong constraints on the number of parameters than can be identified. In dynamical system models, the parameter count grows linearly with population size (for a constant latent dimension), whereas the parameters of a coupled GLM depend quadratically on the number of neurons. Thus, GLMs may have many more parameters, and depend on aggressive regularisation techniques to avoid over-fitting to small datasets.

Here we show that population activity in monkey motor cortex is better fit by a dynamical system model than by a spike-response GLM; and that the dynamical system, but not a GLM of the same temporal resolution, accurately reproduces the temporal structure of cross-correlations in these data.

## 1.2 Comparing dynamical system models with spiking or Gaussian observations

Many studies of population latent variable models assume Gaussian observation noise [8, 17] (but see, e.g. [2, 11, 13, 18]). Given that spikes are discrete events in time, it seems more natural to use a Poisson [10] or other point-process model [19] or, at coarser timescales, a count-process model. However, it is unclear what (if any) advantage such more realistic models confer. For example, Poisson decoding models do not always outperform Gaussian ones [2, 11]. Here, we describe a latent linear dynamical system whose count distribution, when conditioned on all past observations, is Poisson. (the *Poisson linear dynamical system* or PLDS). Using a co-smoothing metric, we show that this (computationally more expensive) count model predicts spike counts in our data better than a Gaussian linear dynamical system (GLDS). The two models give substantially different population spike-count distributions, and the count approach is also more accurate on this measure than either the GLDS or GLM.

## 2 Methods

### 2.1 Dynamical systems with count observations and time-varying mean rates

We first consider the count-process latent dynamical model (PLDS). Denote by $y_{kt}^i$ the observed spike-count of neuron $i \in \{1 \dots q\}$ at time bin $t \in \{1 \dots T\}$ of trial $k \in \{1 \dots N\}$, and by $\mathbf{y}_k = \mathrm{vec}\,(y_{k,i=1:q,t=1:T})$ the $qT \times 1$ vector of all data observed on trial k. Neurons are assumed to be conditionally independent given the low-dimensional latent state $\mathbf{x}_{kt}$ (of dimensionality $p$ with $p < q$). Thus, correlated neural variability arises from variations of this latent population state, and not from direct interaction between neurons. Conditioned on $\mathbf{x}$ and the recent spiking history $\mathbf{s}_t$, the activity of neuron $i$ at time $t$ is given by a Poisson distribution with mean

$$\mathrm{E}[y_{kt}^i | \mathbf{s}_{kt}, \mathbf{x}_{kt}] = \exp\left([C\mathbf{x}_{kt} + d + D\mathbf{s}_{kt}]_i\right), \tag{1}$$

where the $q \times p$ matrix $C$ determines how each neuron is related to the latent state $\mathbf{x}_{kt}$, and the $q$-dimensional vector $d$ controls the mean firing rates of the population. The history term $\mathbf{s}_t$ is a vector of all relevant recent spiking in the population [1, 3, 7, 20]. For example, one choice to model spike refractoriness would set $\mathbf{s}_{kt}$ to the counts at the previous time point $\mathbf{s}_{kt} = y_{k,(t-1)}$, and $D$ to a diagonal matrix of size $q \times q$ with negative diagonal entries. In general, however, $s$ and $D$ may contain entries that reflect temporal dependence on a longer time-scale. However, to maintain the conditional independence of neurons given latent state the matrix $D$ (of size $q \times \dim(s)$) is constrained to have zero values at all entries corresponding to cross-neuron couplings. The exponential nonlinearity ensures that the conditional firing rate of each neuron is positive. Furthermore, while conditioned on the latent state and the recent spiking history the count in each bin is Poisson distributed (hence the model name), samples from the model are not Poisson as they are affected both by variations in the underlying state and the single-neuron history.

We assume that the latent population state $\mathbf{x}_{kt}$ evolves according to driven linear Gaussian dynamics:

$$\mathbf{x}_{k1} \sim \mathcal{N}\,(\mathbf{x}_o, Q_o) \tag{2}$$

$$\mathbf{x}_{k(t+1)} | \mathbf{x}_{kt} \sim \mathcal{N}\,(A\mathbf{x}_{kt} + b_t, Q) \tag{3}$$

Here, $\mathbf{x}_o$ and $Q_o$ denote the average value and the covariance of the initial state $\mathbf{x}_1$ of each trial. The $p \times p$ matrix $A$ specifies the deterministic component of the evolution from one state to the next, and the matrix $Q$ gives the covariance of the innovations that perturb the latent state at each time step. The 'driving inputs' $b_t$, which add to the latent state, allow the model to capture time-varying structure in the firing rates that is consistent across trials. Such time-varying mean firing rates are usually characterised by the *peri-stimulus time histogram* (PSTH), which requires $q \times T$ parameters to estimate for each stimulus. Here, by contrast, time-varying means are captured by the driving inputs into the latent state, and so only $p \times T$ parameters are needed to describe all the PSTHs.

### 2.2 Expectation-Maximisation for the PLDS model

We use an EM algorithm, similar to those described before [10, 11, 12], to learn the parameters $\Theta = \{C, D, d, A, Q, Q_o, \mathbf{x}_o\}$. The E-step requires the posterior distribution $P(\bar{\mathbf{x}}_k | \mathbf{y}_k, \Theta)$ over the latent trajectories $\bar{\mathbf{x}}_k = \mathrm{vec}\,(\mathbf{x}_{k,1:T})$ given the data and our current estimate of the parameters $\Theta$. As this distribution is not available in closed-form, we approximate it by a multivariate Gaussian, $P(\bar{\mathbf{x}}_k | \mathbf{y}_k, \Theta) \approx \mathcal{N}\,(\mu_k, \Sigma_k)$. As $\mathbf{x}_k$ is a vector of length $pT$, so $\mu_k$ and $\Sigma_k$ are of size $pT \times 1$ and $pT \times pT$, respectively. We find the mean $\mu_k$ and the covariance $\Sigma_k$ of this Gaussian via a global Laplace approximation [21], i.e. by maximising the log-posterior $P(\bar{\mathbf{x}}_k, \mathbf{y}_k)$ of each trial over $\mathbf{x}_k$, setting $\mu_k = \mathrm{argmax}_{\bar{\mathbf{x}}} P(\bar{\mathbf{x}} | \mathbf{y}_k, \Theta)$ to be the latent trajectory that achieves this maximum, and $\Sigma_k = -\left(\nabla\nabla_{\bar{\mathbf{x}}} \log P(\bar{\mathbf{x}} | \mathbf{y}_k, \Theta)|_{\bar{\mathbf{x}}=\mu_k}\right)^{-1}$ to be the negative inverse Hessian of the log-posterior at its maximum. The log-posterior on trial k is given by

$$\log P(\bar{\mathbf{x}}_k | \mathbf{y}_k, \Theta) = \mathrm{const} + \sum_{t=1}^{T}\left(\mathbf{y}_{kt}^\top (C\mathbf{x}_{kt} + D\mathbf{s}_{kt} + d) - \sum_{i=1}^{q} \exp\left[C\mathbf{x}_{kt} + D\mathbf{s}_{kt} + d\right]_i\right)$$
$$-\frac{1}{2}(\mathbf{x}_{k1} - \mathbf{x}_o)^\top Q_o^{-1}(\mathbf{x}_{k1} - \mathbf{x}_o) - \frac{1}{2}\sum_{t=1}^{T-1}(\mathbf{x}_{k,t+1} - A\mathbf{x}_{kt} - b_t)^\top Q^{-1}(\mathbf{x}_{k,t+1} - A\mathbf{x}_{kt} - b_t) \tag{4}$$

Log-posteriors of this type are concave and hence unimodal [5, 6], and the Markov structure of the latent dynamics makes it possible to compute a Newton update in $O(T)$ time [22]. Furthermore, it

has previously been observed that the Laplace approximation performs well for similar models with Poisson observations [23]. We checked the quality of the Laplace approximation for our parameter settings by drawing samples from the true posterior in a few cases. The agreement was generally good, with only some minor deviations between the approximated and sampled covariances.

The M-step requires optimisation of the expected joint log-likelihood with respect to the parameters $\Theta$, i.e. $\Theta_{new} = \text{argmax}_{\Theta'} L(\Theta')$ with

$$L(\Theta') = \sum_k \int \left[ \log P(\mathbf{y}_k|\mathbf{x}, \Theta') + \log P(\mathbf{x}|\Theta') \right] \mathcal{N}(\mathbf{x}|\mu_k, \Sigma_k) \, d\mathbf{x}. \tag{5}$$

This integral can be evaluated in closed form, and efficiently optimised over the parameters: $L(\Theta')$ is jointly concave in the parameters $C, d, D$, and the updates with respect to the dynamics parameters $A, Q, Q_o, \mathbf{x}_o$ and the driving inputs $b_t$ can be calculated analytically.

Our use of the Laplace approximation in the E-step breaks the usual guarantee of non-decreasing likelihoods in EM. Furthermore, the full likelihood of the model can only be approximated using sampling techniques [11]. We therefore monitored convergence using the leave-one-neuron-out prediction score [8] that we also used for comparisons with alternative methods (see below): For each trial in the test-set, and for each neuron $i$, we calculate its most likely firing rate given the activity of the other neurons $y_{k,1:T}^{-i}$, and then compared this prediction against the observed activity. If implemented naively, this requires $q$ inferences of the latent state from the activity of $q-1$ neurons. However, this computation can be sped up by an order of magnitude by first finding the most likely state given all neurons, and then performing one Newton-update for each held out neuron from this initial state. While this approximate approach yielded accurate results, we only used it for tracking convergence of the algorithm, not for reporting the results in section 3.1.

### 2.3 Alternative models: Generalised Linear Models and Gaussian dynamical systems

The spike-response GLM models the instantaneous rate of neuron $i$ at time $t$ by a generalised linear form [4] with input covariates representing stimulus (or time) and population spike history:

$$\lambda_{kt}^i = E\left(y_{kt}^i|\mathbf{s}_{kt}\right) = \exp\left([b_t + d + D\mathbf{s}_{kt}]_i\right). \tag{6}$$

The coupling matrix $D$ describes dependence both on the history of firing in the same neuron and on spiking in other neurons, and the $q \times 1$ vectors $b_t$ model time-varying mean firing rates. The parameters are estimated by minimising the negative log-likelihood $L_{dat} = \sum_{kti} \left(y_{kt}^i \log \lambda_{kt}^i - \lambda_{kt}^i\right)$. While equation (6) is similar to the definition of the PLDS model in equation (1), the models differ in their treatment of shared variability: The GLM has no latent state $\mathbf{x}_t$ and so shared variance is modelled through the cross-coupling terms of the matrix $D$, which are set to 0 in the PLDS.

As the number of parameters in the GLM is quadratic in population size, it may be prone to overfitting on small datasets. To improve the generalisation ability of the GLM we added a sparsity-inducing $L_1$ prior on the coupling parameters and a smoothness prior on the PSTH parameters $b_t$, and minimized the (convex) cost function using methods described in [24]:

$$L(b, d, D) = L_{dat} + \eta_1 \sum_{ij} |D_{ij}| + \frac{1}{2\eta_2} \sum_t b_t^\top K_{\eta_3}^{-1} b_t. \tag{7}$$

Here, the regularization parameter $\eta_1$ determines the sparsity of the solution $D$, $\eta_2$ is the prior variance of the smoothing prior, and $K_{\eta_3}(t,s) = \exp\left(-(s-t)^2/\eta_3^2\right)$ is a squared-exponential prior on the time-varying firing rates $b_t$ which ensures their smoothness over time.

It is important to note that GLMs with Poisson conditionals cannot easily be extended to allow for instantaneous couplings between neurons. Suppose that we sought a model whose couplings were only instantaneous, with conditional distributions $y_{it}|\mathbf{y}_{-i,t} \sim \text{Poiss}\left(D_{(i,-i)}\mathbf{y}_{-i,t}\right)$. It can be verified that the model $P(\mathbf{y}) = \frac{1}{Z} \exp\left(y^\top J y\right) / \prod_i y_i!$, which could be regarded as the Poisson equivalent to the Ising model [25], would provide such a structure (as long as $J$ has a zero diagonal). In this model, $P(y_{it}|\mathbf{y}_{-i,t}) \propto \exp(y_{i,t} \sum_{j \neq i} D_{ij} y_{j,t})/y_{i,t}!$. One might imagine that the parameters $J$ could be learnt by maximizing each of the conditional likelihoods over a row of $J$ (effectively maximising the pseudo-likelihood), and one could sample counts by Gibbs sampling, again exploiting the fact that the conditional distributions are all Poisson. However, an obvious prerequisite would be that a $Z$ exists for which the model is normalised. Unfortunately, this becomes impossible as soon as any entry of $J$ is positive. For example, if entry $J_{ij}$ is positive, then we can easily construct a firing

pattern $\mathbf{y}$ for which probabilities diverge. Let the pattern $\mathbf{y}(n)$ have value $n$ at entries $i$ and $j$, and zeros otherwise. Then, for large $n$, we find that $\log P(\mathbf{y}(n)) \propto n^2 J_{ij} - 2\log(n!)$, which is dominated by the quadratic term, and therefore diverges, rendering the model unnormalizeable. Thus, this "Poisson equivalent" of the Ising model cannot model positive interactions between neurons, limiting its value.

The Poisson likelihood of the PLDS requires approximation and is computationally cumbersome. An apparently less veridical alternative would be to model counts as conditionally Gaussian given the latent state. We used the EM algorithm [9] to fit a linear dynamical system model with Gaussian noise and driving inputs [17] (GLDS). In comparison with the Poisson model, the GLDS has an additional set of $q$ parameters corresponding to the variances of the Gaussian observation noise. Finally, we also compared PLDS to Gaussian Process Factor Analysis (GPFA) [8], a Gaussian model in which the latent trajectories are drawn not from a linear dynamical system, but from a more general Gaussian Process with (here) a squared-exponential kernel. We did not include the driving inputs $b_t$ in this model, and used the full model for co-smoothing, i.e. we did not orthogonalise its filters as was done in [8].

We quantified goodness-of-fit using two measures of 'leave-one-neuron-out prediction' accuracy on test data (see [8] for more detail). Each neuron's firing rate was first predicted using the activity of all other neurons on each test trial. For the GLM (but not PLDS), predictions reported were based on the past activity of other neurons, but also used the observed past activity of the neuron being predicted (results exploiting all data from other neurons were similar). Then we calculated the difference between the total variance and the residual variance around this prediction $M_{i,k} = \mathrm{var}(y_{k,1:T}^i) - \mathrm{MSE}(y_{k,1:T}^i, y_{pred})$. Here, the predicted firing rate is a vector of length $T$, and the variance is computed over all times $t = 1, \ldots, T$ in trial $k$. Positive values indicate that prediction is more accurate than a constant prediction equal to the true mean activity of that neuron on that trial. We also constructed a receiver operating characteristic (ROC) for deciding based on the predicted firing rates which bins were likely to contain at least one spike, and measured the area under this curve (AUC) [7, 26]. This measure ranges between $0.5$ and $1$, with a value of $1$ reflecting correct identification of spike-containing bins, even if the predicted number of spikes is incorrect.

## 2.4 Details of neural recordings and choice of parameters

We evaluated the methods described above on multi-electrode recordings from the motor cortex of a behaving monkey. The details of the data are described elsewhere [8]. Briefly, spikes were recorded with a 96-electrode array (Blackrock, Salt Lake City, UT) implanted into motor areas of a rhesus macaque (monkey G) performing a delayed center-out reach task. For the analyses presented here, data came from 108 trials on which the monkey was instructed to reach to one target. We used 1200 ms of data from each trial, from 200ms before target onset until the cue to move was presented. We included 92 units (single and multi-units) with robust delay activity. Spike trains were binned at 10ms which resulted in $8.13\%$ of bins containing at least one spike, and in $0.61\%$ of bins containing more than one spike. For goodness-of fit analyses, we performed 4-fold cross-validation, splitting the data into four non-overlapping test folds with 27 trials each.

For the PLDS model, dimensionality of the latent state varied from 1 to 20. Models either had no direct history-dependence (i.e. $D = 0$), or used spike history mapped to a set of 4 basis functions formed by othogonalising decaying exponentials with time constants $0.1, 10, 20, 40$ms (similar to those used in [1]). The history term $\mathbf{s}_t$ was then obtained by projecting spike counts in the previous 100ms onto each of these functions. The exponential with $0.1$ms time constant effectively covered only the previous time bin and was thus able to model refractoriness. In this case, $D$ was of size $q \times 4q$, with only 4 non-zero elements in each row. For the GLM, we varied the sparsity parameter $\eta_1$ from 0 to 1 (yielding estimates of $D$ that ranged from a dense matrix to entirely 0), and computed prediction performance at each prior setting. After exploratory runs, the parameters of the smoothness prior were set to $\eta_2 = 0.1$ and $\sqrt{\eta_3} = 20$ms.

## 3 Results

### 3.1 Goodness-of-fit of dynamical system models and GLMs

We first compared the goodness-of-fit of PLDS with $p = 5$ latent dimensions against those of GLMs. For all choices of the regularization parameter $\eta_1$ tested, we found that the prediction performance of

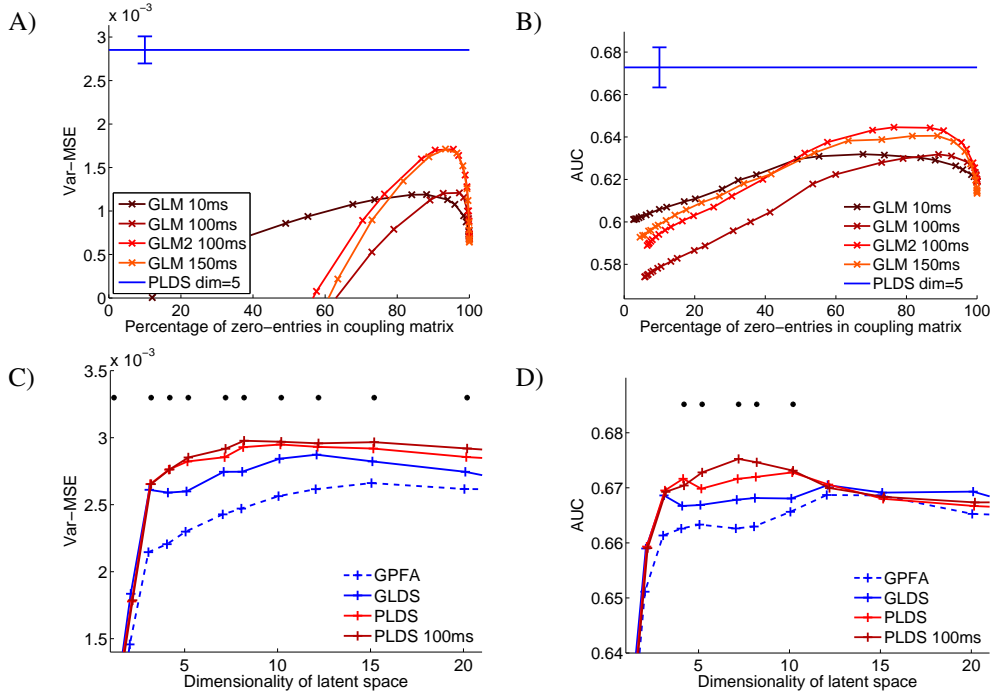

Figure 1: **Quantifying goodness-of-fit. A)** Prediction performance (variance minus mean-squared error on test-set) of various coupled GLMs (10 ms history; 2 variants with 100 ms history; 150 ms history) plotted against sparsity in the filter matrix $D$ generated by different choices of $\eta_1$. For all $\eta_1$, GLM prediction was poorer than that of PLDS with $p = 5$. Error bars on PLDS-performance are standard errors of mean across trials. **B)** As A, measuring performance by area under the ROC-curve (AUC). **C)** Prediction performance of different latent variable models (GPFA, and LDSs with Gaussian, Poisson or history-dependent Poisson noise) on the test-set. Black dots indicate dimensionalities where PLDS with 100ms history is significantly better than GLDS ($p < 0.05$, pairwise comparisons of trials). PLDS outperforms alternatives, and performance plateaus at small latent dimensionalities. **D)** As C, but using AUC to quantify prediction performance. The ordering of the methods (at the optimal dimensionality) is similar, but there is no advantage of PLDS for higher dimensional models.

GLMs was inferior to that of PLDS (Fig. 1A). This was true for GLMs with history terms of length 10ms, 100ms or 150ms (with 1, 4 or 5 basis functions each, which were equivalent to the history functions used for the spiking-history in the dynamical system model, with an additional 80 ms time-constant exponential as the 5th basis function). To ensure that this difference in performance is not due to the GLM over-fitting the terms $b_t$ (which have $q \times T$ parameters for the GLM, but only $p \times T$ parameters for PLDS), we fitted both GLMs and PLDS without those filters. In this case, the prediction performance of both models decreased slightly, but the latent variable models still had substantially better prediction performance.

Our performance metric based on the mean-squared error is sensitive both to the prediction of which bins contain spikes, as well as to how many they contain. To quantify the accuracy with which our models predicted only the absence or presence of spikes, we calculated the area under the curve (AUC) of the receiver operating characteristic [7]. As can be seen in Fig. 1 B the PLDS outperformed the GLMs over all choices of the regularization parameter $\eta_1$.

Next, we investigated a more realistic spiking noise model would further improves the performance of the dynamical system model, and how this would depend on the latent dimensionality d. We therefore compared our models (GPFA, GLDS, PLDS, PLDS with 100ms history) for different choices of the latent dimensionality d. When quantifying prediction performance using the mean-squared error, we found that for all four models, prediction performance on the test-set increased strongly with dimensionality for small dimensions, but plateaued at about 8 to 10 dimensions (see Fig. 1C). Thus, of the models considered here, a low-dimensional latent variable provides the best fit to the data.

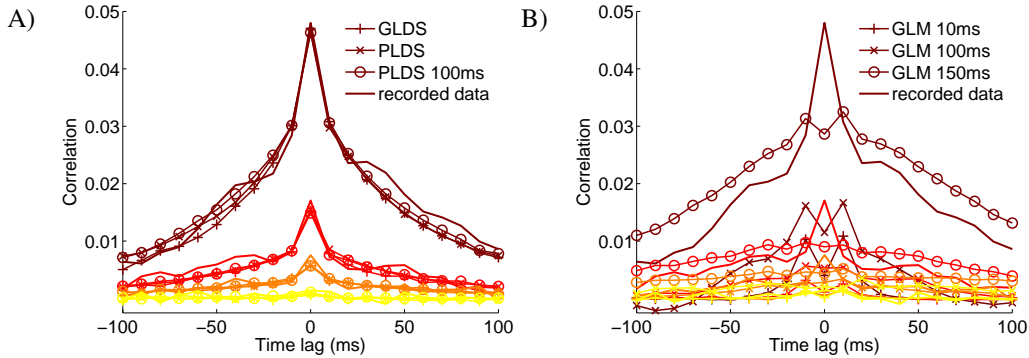

Figure 2: **Temporal structure of cross-correlations. A)** Average temporal cross-correlation in four groups of neurons (color-coded from most to least correlated), and comparison with correlations captured by the dynamical system models with Gaussian, Poisson or history-dependent Poisson noise. All three model correlations agree well with the data. **B)** Comparison of GLMs with differing history-dependence with cortical recordings; the correlations of the models differ markedly from those of the data, and do not have a peak at zero time-lag.

We also found that models with the more realistic spiking noise model (PLDS, and PLDS 100ms) had a small, but consistent performance benefit over the computationally more efficient Gaussian models (GLDS, GPFA). However, for the dataset and comparison considered here (which was based on predicting the mean activity averaged over all possible spiking histories), we only found a small advantage of also adding single-neuron dynamics (i.e. the spike-history filters in $D$) to the spiking noise model. If we compared the models using their ability to predict population activity on the next time-step from the observed population history, single-neuron filters did have an effect. In this prediction task, PLDS with history filters performed best, in particular better than GLMs.

When using AUC rather than mean-squared-error to quantify prediction performance, we found similar results: Low-dimensional models showed best performance, spiking models slightly outperformed Gaussian ones, and adding single-neuron dynamics yielded only a small benefit. In addition, when using AUC, the performance benefit of PLDS over GLDS was smaller, and was significant only at those state-dimensionalities for which overall prediction performance was best. Finally, both GPFA and GLDS at $p = 5$ outperformed all GLMs, both for using AUC and mean-squared-error. Thus, all four of our latent variable models provided superior fits to the dataset than GLMs.

### 3.2 Reproducing the correlations of cortical population activity

In the introduction, we argued that dynamical system models would be more appropriate for capturing the typical temporal structure of cross-neural correlations in cortical multi-cell recordings. We explicitly tested this claim in our cortical recordings. First, we subtracted the time-varying mean firing rate (PSTH) of each neuron to eliminate correlations induced by similarity in mean firing rates. Then, we calculated time-lagged cross-correlations for each pair of neurons, using 10ms bins. For display purposes, we divided neurons into 4 groups (color coded in Fig. 2) according to their total correlation (using summed correlation coefficients with all other neurons), and calculated the average pairwise correlation in each group. Fig. 2A shows the resulting average time-lagged correlations, and shows that both dynamical system models accurately capture this aspect of the correlation structure of the data. In contrast Fig. 2B shows that the temporal correlations of the GLM differ markedly from the real data[1]. As mentioned before, this GLM is also fit at 10ms resolution, leaving open the possibility that fitting it at a finer temporal resolution would yield samples which more closely reflect the recorded correlations.

### 3.3 Reproducing the distribution of spike-counts across the population

In the above, we showed that the PLDS model outperforms both Gaussian models and GLMs with respect to our performance-metric, and that samples from both dynamical systems accurately capture the temporal correlation structure of the data. Finally, we looked at an aggregate measure of

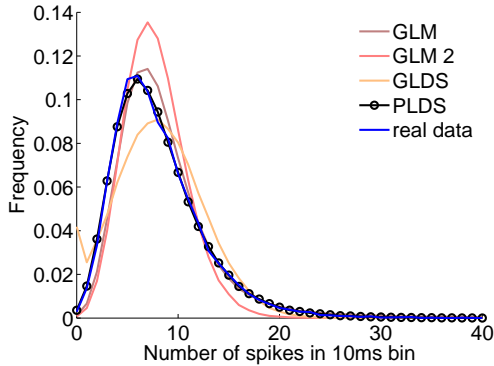

Figure 3: **Modeling population spike counts.** Distribution of the population spike counts, and comparison with distributions from PLDS, GLDS and two versions of the GLM with 150ms history dependence (GLM with no regularization, GLM2 with optimal sparsity).

population activity, namely the distribution of population spike counts, i.e. the distribution of the total number of spikes across the population per time bin. This distribution is influenced both by the single-neuron spike-count distributions and second- and higher-order correlations across neurons. Fig. 3 shows that the PLDS model accurately reproduces the spike-count distribution in the data, whereas the other two models do not. The GLDS model underestimates the frequency of high spike counts, despite accurately matching both the mean and the variance of the distribution. For the GLM (using 150ms history, and either no regularization or optimal regularization), the frequency of rare events is either over- or under-estimated. This could be further indication that the GLM does not fully capture the fact that variability is shared across many cells in the population.

## 4   Discussion

We explored a statistical model of cortical population recordings based on a latent dynamical system with count-process observations. We argued that such a model provides a more natural modeling choice than coupled spike-response GLMs for cortical array-recordings; and indeed, this model did fit motor-cortical multi-unit recording better, and more faithfully reproduced the temporal structure of cross-neural correlations. GLMs have many attractive properties, and given the flexibility of the model class, it is impossible to rule out that *some* coupled GLM with finer temporal resolution, possibly nonlinear history dependencies and cleverly chosen regularization would yield better cross-validation performance. We here argued that latent variable models yield a more appropriate model of cross-neural correlations with zero-lag peaks: In GLMs, one has to use a fine discretization of the time-axis (which can be computationally intensive) or work in continuous time to achieve this. Thus, they might constitute good point-process models at fine time-scales, but arguably not the right count-process model to model neural recordings at coarser time-scales.

We also showed that a model with count-process observations yields better fits to our data than ones with a Gaussian noise model, and that it has a more realistic distribution of population spike counts. Given that spiking data is discrete and therefore non-Gaussian, this might not seem surprising. However, it is important to note that the Gaussian model has free parameters for the single-neuron variability, whereas the conditional variance of the Poisson model is constrained to equal the mean. For data in which this assumption is invalid, use of other count models, such as a negative binomial distribution, might be more appropriate. In addition, fitting the PLDS model requires simplifying approximations, and these approximations could offset any gain in prediction performance. As measured by our co-smoothing metrics, the performance advantage of our count-process over the Gaussian noise model was small, and the question of whether this advantage would justify the considerable additional computational cost of the count-process model will depend on the application at hand. In addition, any comparison of statistical models depends on the data used, as different methods are appropriate for datasets with different properties. For the recordings we considered here, a dynamical system model with count-process observations worked best, but there will be datasets for which either GLMs, or GLDS or GPFA provide the most appropriate model. Finally, the choice of the most appropriate model depends on the analysis or prediction question of interest. While we used a co-smoothing metric to quantify model performance, different models might be more suitable for decoding reaching movements from population activity [11], or inferring the underlying anatomical connectivity from extracellular recordings.

## Acknowledgements

We acknowledge support from the Gatsby Charitable Foundation, an EU Marie Curie Fellowship to JHM, EPSRC EP/H019472/1 to JPC, the Defense Advanced Research Projects Agency (DARPA) through "Reorganization and Plasticity to Accelerate Injury Recovery (REPAIR; N66001-10-C-2010)", NIH CRCNS R01-NS054283 to KVS and MS as well as NIH Pioneer 1DP1OD006409 to KVS.

## Footnotes

[1]We used $\eta_1 = 0$, i.e. no regularization for this figure, results with $\eta_1$ optimized for prediction performance vastly underestimate correlations in the data.

## References

[1] J. W. Pillow, J. Shlens, L. Paninski, A. Sher, A. M. Litke, E. J. Chichilnisky, and E. P. Simoncelli. Spatio-temporal correlations and visual signalling in a complete neuronal population. *Nature*, 454(7207):995–999, 2008.

[2] G. Santhanam, B. M. Yu, V. Gilja, S. I. Ryu, A. Afshar, M. Sahani, and K. V. Shenoy. Factor-analysis methods for higher-performance neural prostheses. *J Neurophysiol*, 102(2):1315–1330, 2009.

[3] E.S. Chornoboy, L.P. Schramm, and A.F. Karr. Maximum likelihood identification of neural point process systems. *Biological Cybernetics*, 59(4):265–275, 1988.

[4] P. McCulloch and J. Nelder. Generalized linear models. *Chapman and Hall, London*, 1989.

[5] L. Paninski. Maximum likelihood estimation of cascade point-process neural encoding models. *Network*, 15(4):243–262, 2004.

[6] S.P. Boyd and L. Vandenberghe. *Convex optimization*. Cambridge Univ Press, 2004.

[7] W. Truccolo, L. R. Hochberg, and J. P. Donoghue. Collective dynamics in human and monkey sensori-motor cortex: predicting single neuron spikes. *Nat Neurosci*, 13(1):105–111, 2010.

[8] B. M. Yu, J. P. Cunningham, G. Santhanam, S. I. Ryu, K. V. Shenoy, and M. Sahani. Gaussian-process factor analysis for low-dimensional single-trial analysis of neural population activity. *J Neurophysiol*, 102(1):614–635, 2009.

[9] S. Roweis and Z. Ghahramani. A unifying review of linear gaussian models. *Neural Comput*, 11(2):305–345, 1999 Feb 15.

[10] A. C. Smith and E. N. Brown. Estimating a state-space model from point process observations. *Neural Comput*, 15(5):965–91, 2003.

[11] V. Lawhern, W. Wu, N. Hatsopoulos, and L. Paninski. Population decoding of motor cortical activity using a generalized linear model with hidden states. *J Neurosci Methods*, 189(2):267–280, 2010.

[12] J.E. Kulkarni and L. Paninski. Common-input models for multiple neural spike-train data. *Network: Computation in Neural Systems*, 18(4):375–407, 2007.

[13] M. Vidne, Y. Ahmadian, J. Shlens, J.W. Pillow, J Kulkarni, E. J. Chichilnisky, E. P. Simoncelli, and L Paninski. A common-input model of a complete network of ganglion cells in the primate retina. In *Computational and Systems Neuroscience*, 2010.

[14] M. M. Churchland, B. M. Yu, M. Sahani, and K. V. Shenoy. Techniques for extracting single-trial activity patterns from large-scale neural recordings. *Current Opinion in Neurobiology*, 17(5):609–618, 2007.

[15] D. Y. Tso, C. D. Gilbert, and T. N. Wiesel. Relationships between horizontal interactions and functional architecture in cat striate cortex revealed by cross-correlation analysis. *J Neurosci*, 6(4):1160–1170, 1986.

[16] A. Jackson, V. J. Gee, S. N. Baker, and R. N. Lemon. Synchrony between neurons with similar muscle fields in monkey motor cortex. *Neuron*, 38(1):115–125, 2003.

[17] W. Wu, Y. Gao, E. Bienenstock, J.P. Donoghue, and M.J. Black. Bayesian population decoding of motor cortical activity using a kalman filter. *Neural Comput*, 18(1):80–118, 2006.

[18] B. Yu, A. Afshar, G. Santhanam, S.I. Ryu, K. Shenoy, and M. Sahani. Extracting dynamical structure embedded in neural activity. In *Advances in Neural Information Processing Systems*, volume 18, pages 1545–1552. MIT Press, Cambridge, 2006.

[19] J.P. Cunningham, B.M. Yu, K.V. Shenoy, and M. Sahani. Inferring neural firing rates from spike trains using gaussian processes. *Advances in neural information processing systems*, 20:329–336, 2008.

[20] U. T. Eden, L. M. Frank, R. Barbieri, V. Solo, and E. N. Brown. Dynamic analysis of neural encoding by point process adaptive filtering. *Neural Comput*, 16(5):971–98, 2004.

[21] B. Yu, J. Cunningham, K. Shenoy, and M. Sahani. Neural decoding of movements: From linear to nonlinear trajectory models. In *Neural Information Processing*, pages 586–595. Springer, 2008.

[22] L. Paninski, Y. Ahmadian, D. G. Ferreira, S. Koyama, K. Rahnama Rad, M. Vidne, J. Vogelstein, and W. Wu. A new look at state-space models for neural data. *J Comput Neurosci*, 29(1-2):107–126, 2010.

[23] Y. Ahmadian, J. W. Pillow, and L. Paninski. Efficient markov chain monte carlo methods for decoding neural spike trains. *Neural Comput*, 23(1):46–96, 2011.

[24] G. Andrew and J. Gao. Scalable training of l 1-regularized log-linear models. In *Proceedings of the 24th international conference on Machine learning*, pages 33–40. ACM, 2007.

[25] E. Schneidman, M. J. 2nd Berry, R. Segev, and W. Bialek. Weak pairwise correlations imply strongly correlated network states in a neural population. *Nature*, 440(7087):1007–12, 2006.

[26] T.D. Wickens. *Elementary Signal Detection Theory*. Oxford University Press, 2002.

